# Learning to classify complex patterns using a VLSI network of spiking neurons

**Srinjoy Mitra**[†], **Giacomo Indiveri**[†] **and Stefano Fusi** [†∇]
[†]Institute of Neuroinformatics, UZH|ETH, Zurich
[∇]Center for Theoretical Neuroscience, Columbia University, New York
`srinjoy|giacomo|fusi@ini.phys.ethz.ch`

## Abstract

We propose a compact, low power VLSI network of spiking neurons which can learn to classify complex patterns of mean firing rates on–line and in real–time. The network of integrate-and-fire neurons is connected by bistable synapses that can change their weight using a local spike–based plasticity mechanism. Learning is supervised by a teacher which provides an extra input to the output neurons during training. The synaptic weights are updated only if the current generated by the plastic synapses does not match the output desired by the teacher (as in the perceptron learning rule). We present experimental results that demonstrate how this VLSI network is able to robustly classify uncorrelated linearly separable spatial patterns of mean firing rates.

## 1 Introduction

Spike driven synaptic plasticity mechanisms have been thoroughly investigated in recent years to solve two important problems of learning: 1) how to modify the synapses in order to generate new memories 2) how to protect old memories against the passage of time, and the overwriting of new memories by ongoing activity. Temporal patterns of spikes can be encoded with spike-timing dependent plasticity (STDP) mechanisms (*e.g.* see [1, 2]). However, STDP in its simplest form is not suitable for learning patterns of mean firing rates [3], and most of the proposed STDP learning algorithms solved the problems of memory encoding and memory preservation only for relatively simple patterns of mean firing rates.

Recently a new model of stochastic spike-driven synaptic plasticity has been proposed [4] that is very effective in protecting old learned memories, and captures the rich phenomenology observed in neurophysiological experiments on synaptic plasticity, including STDP protocols. It has been shown that networks of spiking neurons that use this synaptic plasticity model can learn to classify complex patterns of spike trains ranging from stimuli generated by auditory/vision sensors to images of handwritten digits from the MNIST database [4]. Here we describe a neuromorphic VLSI implementation of this spike-driven synaptic plasticity model and present classification experiments using the VLSI device that validate the model's implementation. The silicon neurons and synapses inside the chip are implemented using full custom hybrid analog/digital circuits, and the network's spikes are received in input and transmitted in output using asynchronous digital circuits. Each spike is represented as an *Address-Event*, where the address encodes either the source neuron or the destination synapse. This device is part of an increasing collection of spike-based computing chips that have been recently developed within the framework of Address-Event Representation (AER) systems [5, 6]. There are even multiple implementations of the same spike-driven plasticity model being investigated in parallel [7, 8]. The focus of this paper is to show that the VLSI device proposed here can successfully classify complex patterns of spike trains, producing results that are in accordance with the theoretical predictions.

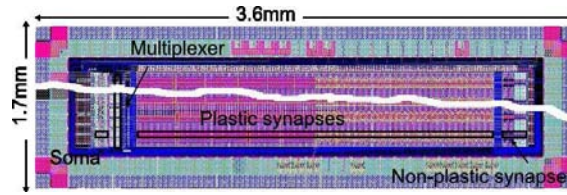

Figure 1: Layout of a test chip comprising a network of I&F neurons and plastic synapses. The placement of a single neuron along with its synapses is highlighted in the top part of the figure. Other highlighted circuits are described in the test.

In Section 2 we describe the main features of the spike-based plasticity model and show how they are well suited for future scaled CMOS VLSI technologies; in Section 3 we characterize the functionality of the spike-based learning circuits; in Section 4 we show control experiments on the learning properties of the VLSI network; and in Section 5 we present experimental results on complex patterns of mean firing rates. In Section 6 we present the concluding remarks and point out future outlooks and potential applications of this system.

## 2   Implementation of the spike-based plasticity mechanism

Physical implementations of long lasting memories, either biological or electronic, are confronted with two hard limits: the synaptic weights are bounded (they cannot grow indefinitely or become negative), and the resolution of the synapse is limited (*i.e.* the synaptic weight cannot have an infinite number of states). These constraints, usually ignored by the vast majority of software models, have strong impact on the classification performance of the network, and on its memory storage capacity. It has been demonstrated that the number of random uncorrelated patterns $p$ which can be classified or stored in a network of neurons connected by bounded synapses grows only logarithmically with the number of synapses [9]. In addition, if each synapse has a $n$ stable states (*i.e.* its weight has to traverse $n$ states to go from the lower bound to the upped bound), then the number of patterns $p$ can grow quadratically $n$. However, this can happen only in unrealistic scenarios, where fine tuning of the network's parameters is allowed. In more realistic scenarios where there are inhomogeneities and variability (as is the case for biology and silicon) $p$ is largely independent of $n$ [9].

Therefore, an efficient strategy for implementing long lasting memories in VLSI networks of spiking neurons is to use a large number of synapses with only two stable states (*i.e.* $n = 2$), and to modify their weights in a stochastic manner, with a small probability. This slows down the learning process, but has the positive effect of protecting previously stored memories from being overwritten. Using this strategy we can build large networks of spiking neurons with very compact learning circuits (*e.g.* that do not require local Analog-to-Digital Converters or floating gate cells for storing weight values). By construction, these types of devices operate in a massively parallel fashion and are fault-tolerant: even if a considerable fraction of the synaptic circuits is faulty due to fabrication problems, the overall functionality of the chip is not compromised. This can be a very favorable property in view of the potential problems of future scaled VLSI processes.

The VLSI test chip used to carry out classification experiments implementing such strategy is shown in Fig. 1. The chip comprises 16 low-power integrate-and-fire (I&F) neurons [5] and 2048 dynamic synapses. It was fabricated using a standard $0.35\mu$m CMOS technology, and occupies an area of $6.1mm^2$ . We use an AER communication infrastructure that allows the chip to receive and transmit asynchronous events (spikes) off-chip to a workstation (for data logging and prototyping) and/or to other neuromorphic event-based devices [10]. An on-chip multiplexer can be used to reconfigure the neuron's internal dendritic tree connectivity. A single neuron can be connected to 128, 256, 512 or 1024 synapses. Depending on the multiplexer state the number of active neurons decrease from 16 to 2. In this work we configured the chip to use all 16 neurons with 128 synapses per neuron. The synapses are divided into different functional blocks: 4 are excitatory with fixed (externally adjustable) weights, 4 inhibitory and 120 excitatory with local learning circuits.

Every silicon neuron in the chip can be used as a classifier that separates the input patterns into two categories. During training, the patterns to be classified are presented to the pre-synaptic synapses,

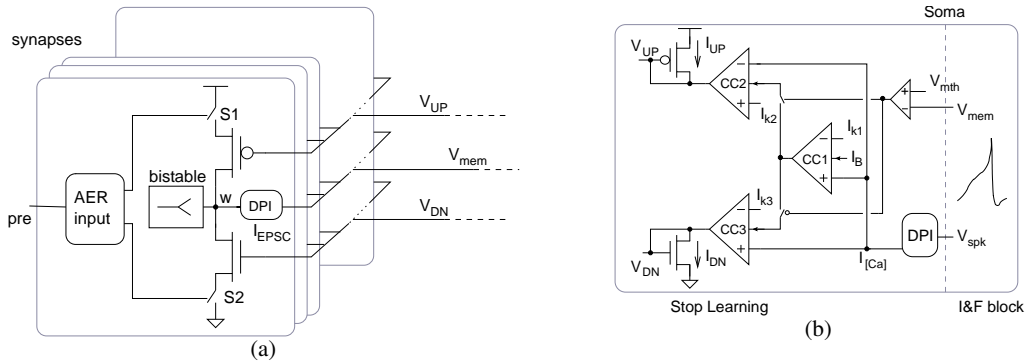

Figure 2: (a) Plastic synapse circuits belonging to the neuron's dendritic tree. The synaptic weight node $w$ is modified when there is a pre-synaptic input (*i.e.* when $S1$ and $S2$ are on) depending on the values of $V_{UP}$ and $V_{DN}$. In parallel, the bistable circuit slowly drives the node $w$ toward either of its two stable states depending on its amplitude. The DPI is a pulse integrator circuit that produces an Excitatory Post-Synaptic Current ($I_{EPSC}$), with an amplitude that depends on the synaptic weight $w$. (b) Neuron's "soma" block diagram with stop-learning module. It comprises a low-power I&F neuron block, a DPI integrator, a voltage comparator and a three current comparators($CC$). Winner-take-all (WTA) circuits are used as current comparators that set the output to be either the bias current $I_B$, or zero. The voltage comparator enables either the $I_{UP}$ or the $I_{DN}$ block, depending on the value of $V_{mem}$ with respect to $V_{mth}$. The voltages $V_{UP}$ and $V_{DN}$ are used to broadcast the values of $I_{UP}$ and $I_{DN}$ to the neuron's dendritic tree.

in parallel with a teacher signal that represents the desired response. The post-synaptic neuron responds with an activity that is proportional to its net input current, generated by the input pattern weighted by the learned synaptic efficacies, and by the teacher signal. If the neuron's mean activity is in accordance with the teacher signal (typically either very high or very low), then the output neuron produces the correct response. In this case the the synapses should not be updated. Otherwise, the synapses are updated at the time of arrival of the (Poisson distributed) input spikes, and eventually make a transition to one of the two stable states. Such stochasticity, in addition to the 'stop-learning' mechanism which prevents the synapses from being modified when the output is correct, allows each neuron to classify a wide class of highly correlated, linearly separable patterns. Furthermore, by using more than one neuron per class, it is possible to classify also complex non-linearly separable patterns [4].

## 3 The VLSI learning circuits

The learning circuits are responsible for locally updating the synaptic weights with the spike-based learning rule proposed in [4].

Upon the arrival of a pre-synaptic spike (an address-event), the plastic synapse circuit updates its weight $w$ according to the spike-driven learning rule. The synapse then produces an Excitatory Post-Synaptic Current (EPSC) with an amplitude proportional to its weight, and with an exponential time course that can be set to last from microseconds to several hundreds of milliseconds [11]. The EPSC currents of all synapses afferent to the target neuron are summed into the neuron's membrane capacitance, and eventually the I&F neuron's membrane potential exceeds a threshold and the circuit generates an output spike. As prescribed by the model of [4], the post-synaptic neuron's membrane potential, together with its mean firing rate are used to determine the weight change values $\Delta w$. These weight change values are expressed in the chip as subthreshold currents. Specifically, the signal that triggers positive weight updates is represented by an $I_{UP}$ current, and the signal that triggers weight decreases if represented by the $I_{DN}$ current.

The weight updates are performed locally at each synapse, in a *pre-synaptic weight update module*, while the $\Delta w$ values are computed globally (for each neuron), in a *post-synaptic weight control module*.

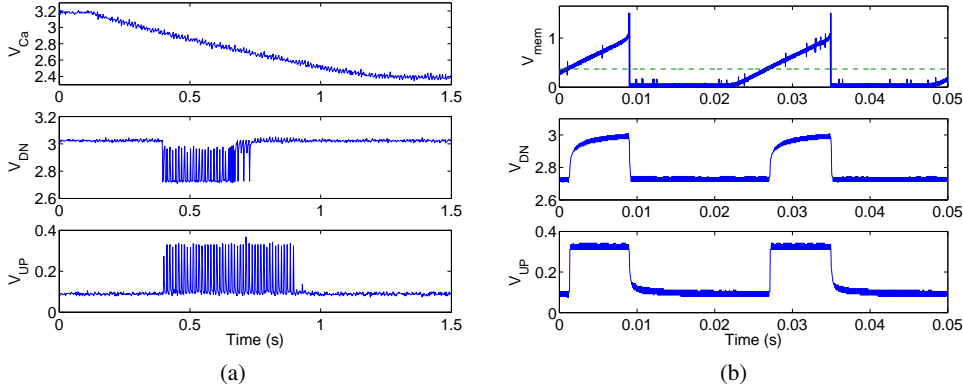

Figure 3: Post-synaptic circuit data. (a) State of the $V_{UP}$ and $V_{DN}$ voltages as a function of the calcium concentration voltage $V_{Ca}$. (b) State of the $V_{UP}$ and $V_{DN}$ voltages as function of the membrane potential $V_{mem}$. This data corresponds to a zoomed-version of the data shown in (a) for $V_{Ca} \approx 2.8V$.

## 3.1 Pre-synaptic weight-update module

This module, shown in Fig. 2(a), comprises four main blocks: an input AER interfacing circuit [12], a bistable weight refresh circuit, a weight update circuit and a log-domain current-mode integrator, dubbed the "diff-pair integrator" (DPI) circuit, and fully characterized in [11]. Upon the arrival of an input event (pre-synaptic spike), the asynchronous AER interfacing circuits produce output pulses that activate switches $S1$ and $S2$. Depending on the values of $I_{UP}$ and $I_{DN}$, mirrored from the post-synaptic weight control module, the node $w$ charges up, discharges toward ground, or does not get updated. The same input event activates the DPI circuit that produces an EPSC current ($I_{EPSC}$) with an amplitude that depends on the synaptic weight value $w$. In parallel, the bistable weight refresh circuit slowly drives $w$ toward one of two stable states depending on whether it is higher or lower than a set threshold value. The two stable states are global analog parameters, set by external bias voltages.

## 3.2 Post-synaptic weight control module

This module is responsible for generating the two global signals $V_{UP}$ and $V_{DN}$, mirrored to all synapses belonging to the same dendritic tree. Post-synaptic spikes ($V_{spk}$), generated in the soma are integrated by an other instance of the DPI circuit to produce a current $I_{Ca}$ proportional to the neuron's average spiking activity. This current is compared to three threshold values, $I_{k1}$, $I_{k2}$, and $I_{k3}$ of Fig. 2(b), using three current-mode winner-take-all circuits [13]. In parallel, the instantaneous value of the neuron's membrane potential $V_{mem}$ is compared to the threshold $V_{mth}$ (see Fig. 2(b)). The values of $I_{UP}$ and $I_{DN}$ depend on the state of the neuron's membrane potential and its average frequency. Specifically, if $I_{k1} < I_{Ca} < I_{k3}$ and $V_{mem} > V_{mth}$, then $I_{UP} = I_B$. If $I_{k1} < I_{Ca} < I_{k2}$ and $V_{mem} < V_{mth}$, then $I_{DN} = I_B$. Otherwise both $I_{UP}$, and $I_{DN}$ are null.

To characterize these circuits we injected a step current in the neuron, produced a regular output mean firing rate, and measured the voltages $V_{Ca}$, $V_{UP}$, and $V_{DN}$ (see Fig. 3(a)). $V_{Ca}$ is the gate voltage of the P-FET transistor producing $I_{Ca}$, while $V_{DN}$, $V_{UP}$ are the gate voltages of the P- and N-FET transistors mirroring $I_{DN}$ and $I_{UP}$ respectively (Fig. 2(a)). The neuron's spikes are integrated and the output current $I_{Ca}$ increases with an exponential profile over time ($V_{Ca}$ decreases accordingly over time, as shown in Fig. 3(a)). The steady-state asymptotic value depends on the average input frequency, as well as the circuit's bias parameters [11]. As $I_{Ca}$ becomes larger than the first threshold $I_{k1}$ ($V_{Ca}$ decreases below the corresponding threshold voltage) both $V_{UP}$ and $V_{DN}$ are activated. When $I_{Ca}$ becomes larger than the second threshold $I_{k2}$ the $V_{DN}$ signal is deactivated, and finally as $I_{Ca}$ becomes larger than the third threshold $I_{k3}$, also the $V_{UP}$ signal is switched off. The small $\sim$ 300mV changes in $V_{UP}$ and $V_{DN}$ produce subthreshold currents ($I_{UP}$ and $I_{DN}$) that are mirrored to the synapses (Fig. 2(a)). In Fig. 3(b) the $V_{DN}$ and $V_{UP}$ signals are zoomed in along with the membrane potential of the post-synaptic neuron ($V_{mem}$), for values of $V_{Ca} \sim 2.8V$. Depending on the state of

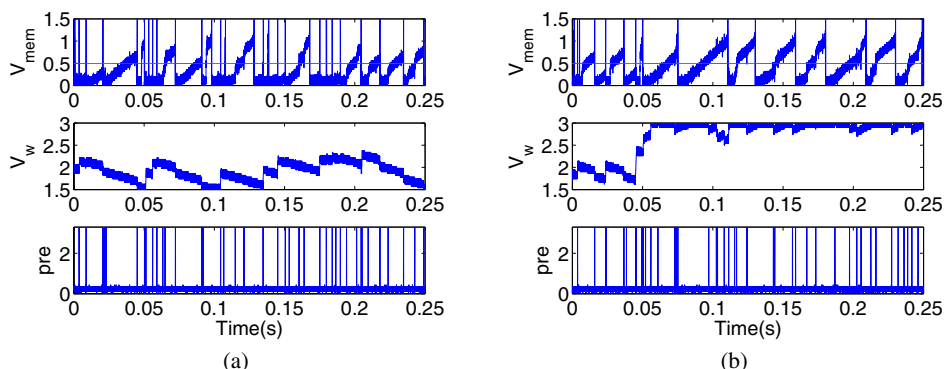

Figure 4: Stochastic synaptic LTP transition: in both sub-figures the non-plastic synapse is stimulated with Poisson distributed spikes at a rate of 250Hz, making the post-synaptic neuron fire at approximately 80Hz; and the plastic synapse is stimulated with Poisson distributed spike trains of 100Hz. (a) The updates in the synaptic weight did not produce any LTP transition during the 250ms stimulus presentation. (b) The updates in the synaptic weight produced an LTP transition that remains consolidated.

$V_{mem}$, the signals $V_{UP}$ and $V_{DN}$ are activated or inactivated. When not null, currents $I_{UP}$ and $I_{DN}$ are complementary in nature: only one of the two is equal to $I_B$.

## 4  Stochastic plasticity

To characterize the stochastic nature of the weight update process we stimulated the neuron's plastic synapses with Poisson distributed spike trains. When any irregular spike train is used as a pre-synaptic input, the synaptic weight voltage crosses the synapse bistability threshold in a stochastic manner, and the probability of crossing the threshold depends on the input's mean frequency. Therefore Long Term Potentiation (LTP) or Long Term Depression (LTD) occur stochastically even when the mean firing rates of the input and the output are always the same. In Fig. 4 we show two instances of a learning experiment in which the mean input firing rate (bottom row) was 100Hz, and the mean output firing rate (top row) was 80Hz. Although these frequencies were the same for both experiments, LTP occurred only in one of the two cases (compare synaptic weight changes in middle row of both panels). In this experiment we set the efficacy of the "high" state of all plastic synapses to a relatively low value. In this way the neuron's mean output firing rate depends primarily on the teacher signal, irrespective of the states of plastic synapses.

One essential feature of this learning rule is the non-monotonicity of both the LTP/LTD probabilities as a function of the post-synaptic firing frequency $v_{post}$ [4]. Such a non-monotonicity is essential to slow down and eventually stop-learning when $v_{post}$ is very high or very low (indicating that the learned synaptic weights are already correctly classifying the input pattern). In Fig. 5 we show experimental results where we measured the LTP and LTD transitions of 60 synapses over 20 training sessions: for the LTD case (top row) we initialized the synapses to a high state (white pixel) and plotted a black pixel if its final state was low, at the end of the training session. The transitions (white to black) are random in nature and occur with a probability that first increases and then decreases with $v_{post}$. An analogous experiment was done for the LTP transitions (bottom row), but with complementary settings (the initial state was set to a low value). In Fig. 5(b) we plot the LTD (top row) and LTP (bottom row) probabilities measured for a single synapse. The shape of these curves can be modified by acting on the post-synaptic weight control module bias parameters such as $I_{k1-k3}$, or $I_B$.

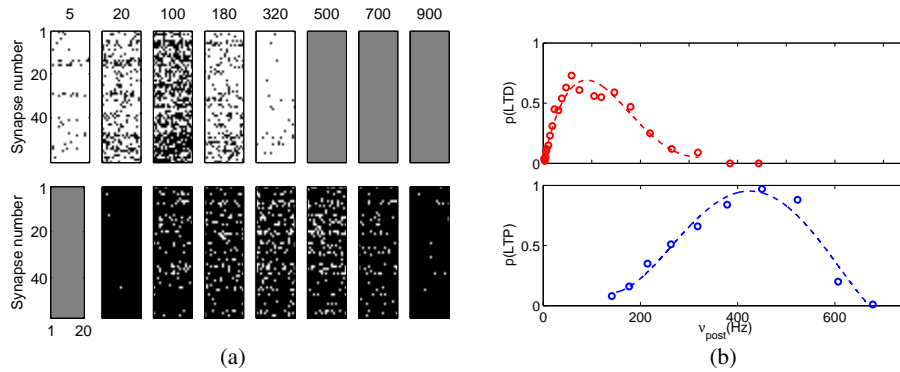

(a)
(b)

Figure 5: (a) LTD and LTP transitions of 60 synapses measured across 20 trials, for different values of post-synaptic frequency $v_{post}$ (top label on each panel). Each black pixel represents a low synaptic state, and white pixel a high one. On x-axis of each panel we plot the trial number (1 to 20) and y-axis shows the state of the synapses at the end of each trial. In the top row we show the LTD transitions that occur after initializing all the synapses to high state. In the bottom row we show the LTP transition that occur after initializing the synapses to low state. The transitions are stochastic and the LTP/LTD probabilities peak at different frequencies before falling down at higher $v_{post}$ validating the stop-learning algorithm. No data was taken for the gray panels. (b) Transition probabilities measured for a single synapse as a function $v_{post}$. The transition probabilities can be reduced by decreasing the value of $I_B$. The probability peaks can also be modified by changing the biases that set $I_{k1-k3}$. (Fig. 2(b))

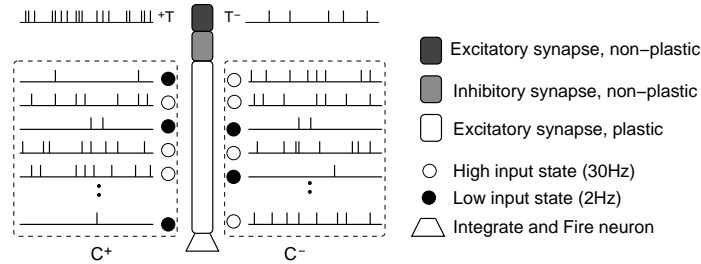

Figure 6: A typical training scenario with 2 random binary spatial patterns. High and low inputs are encoded with generate Poisson spike trains with mean frequencies of 30Hz and 2Hz respectively. Binary patterns are assigned to the $C^+$ or $C^-$ class arbitrarily. During training patterns belonging to the $C^+$ class are combined with a $T^+$ (teacher) input spike train of with 250Hz mean firing rate. Similarly, patterns belonging to the $C^-$ class are combined with a $T^-$ spike train of 20Hz mean firing rate. New Poisson distributed spike trains are generated for each training iterations.

## 5   Classification of random spatial patterns

In order to evaluate the chip's classification ability, we used spatial binary patterns of activity, randomly generated (see Fig. 6). The neuron's plastic synapses were stimulated with Poisson spike trains of either high (30Hz) or low (2Hz) mean firing rates. The high/low binary state of the input was chosen randomly, and the number of synapses used was 60. Each 60-input binary pattern was then randomly assigned to either a $C^+$ or a $C^-$ class.

During training, spatial patterns belonging to the $C^+$ class are presented to the neuron in conjunction with a $T^+$ teacher signal (*i.e.* a 250Hz Poisson spike train). Conversely patterns belonging to the $C^-$ class are combined with a $T^-$ teacher signal of 20Hz. The $T^+$ and $T^-$ spike trains are presented to the neuron's non-plastic synapses. Training sessions with $C^+$ and $C^-$ patterns are interleaved in a random order, for 50 iterations. Each stimulus presentation lasted 500ms, with new Poisson distributions generated at each training session.

After training, the neuron is tested to see if it can correctly distinguish between patterns belonging to the two classes $C^+$ and $C^-$. The binary patterns used during training are presented to the neuron without the teacher signal, and the neuron's mean firing rate is measured. In Fig. 7(a) we plot the responses of two neurons labeled neuron-A and neuron-B. Neuron-A was trained to produce a high output firing rate in response to patterns belonging to class $C^+$, while neuron-B was trained to respond to patterns belonging to class $C^-$. As shown, a single threshold (*e.g.* at 20Hz) is enough to classify the output in $C^+$ (high frequency) and $C^-$ (low frequency) class.

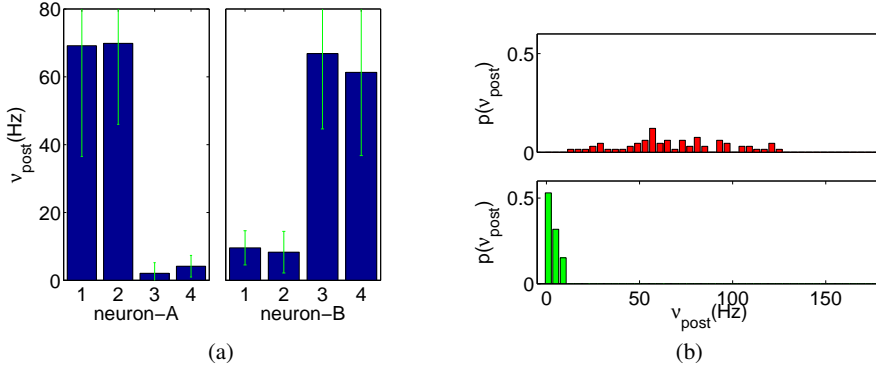

Figure 7: Classification results, after training on 4 patterns. (a) Mean output frequencies of neurons trained to recognize class $C^+$ patterns (Neuron-A), and class $C^-$ patterns (Neuron-B). Patterns $1, 2$ belong to class $C^+$, while patterns $3, 4$ belong to class $C^-$. (b) Output frequency probability distribution, for all $C^+$ patterns (top) and $C^-$ patterns (bottom) computed over 20 independent experiments.

Fig. 7(b) shows the probability distribution of post-synaptic frequencies (of neuron-A) over different classification experiments, each done with new sets of random spatial patterns.

To quantify the chip's classification behavior statistically, we employed a Receiver Operating Characteristics (ROC) analysis [14]. Figure 8(a) shows the area under the ROC curve (AUC) plotted on y-axis for increasing number of patterns. An AUC magnitude of 1 represents 100% correct classification while 0.5 represents chance level. In Fig. 8(b) the storage capacity (p) –expressed as the number of patterns with AUC larger than 0.75– is plotted against the number of synapses $N$. The top and bottom traces show the theoretical predictions from [3], with (p$\propto 2\sqrt{N}$) and without (p$\propto \sqrt{N}$) the *stop learning* condition, respectively. The performance of the VLSI system with 20, 40 and 60 synapses and the stop-learning condition lie within the two theoretical curves.

# 6 Conclusions

We implemented in a neuromorphic VLSI device a recently proposed spike-driven synaptic plasticity model that can classify complex patterns of spike trains [4]. We presented results from the VLSI chip that demonstrate the correct functionality of the spike-based learning circuits, and performed classification experiments of random uncorrelated binary patterns, that confirm the theoretical predictions. Additional experiments have demonstrated that the chip can be applied to the classification of correlated spatial patterns of mean firing rates and as well [15]. To our knowledge, the classification performance achieved with this chip has not yet been reported for any other silicon system. These results show that the device tested can perform real-time classification of sequences of spikes, and is therefore an ideal computational block for adaptive neuromorphic sensory-motor systems and brain-machine interfaces.

## Acknowledgment

This work was supported by the Swiss National Science Foundation grant no. PP00A106556, the ETH grant no. TH02017404, and by the EU grants ALAVLSI (IST-2001-38099) and DAISY (FP6-2005-015803).

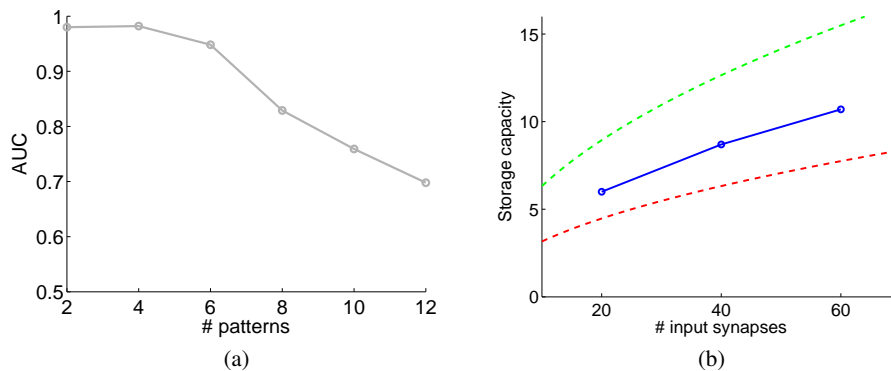

<div align="center">(a)          (b)</div>

Figure 8: (a). Area under ROC curve (AUC) measured by performing 50 classification experiments. (b) Storage capacity (number of patterns with AUC value $\geq 0.75$) as a function of the number of plastic synapses used. The solid line represents the data obtained from chip, while top and bottom traces represent the theoretical predictions with and without the *stop learning* condition.

# References

[1] R. Gütig and H. Sompolinsky. The tempotron: a neuron that learns spike timing–based decisions. *Nature Neuroscience*, 9:420–428, 2006.

[2] R.A. Legenstein, C. Näger, and W. Maass. What can a neuron learn with spike-timing-dependent plasticity? *Neural Computation*, 17(11):2337–2382, 2005.

[3] S. Fusi and W. Senn. Eluding oblivion with smart stochastic selection of synaptic updates. *Chaos, An Interdisciplinary Journal of Nonlinear Science*, 16(026112):1–11, 2006.

[4] J. Brader, W. Senn, and S. Fusi. Learning real world stimuli in a neural network with spike-driven synaptic dynamics. *Neural Computation*, 2007. (In press).

[5] G. Indiveri, E. Chicca, and R. Douglas. A VLSI array of low-power spiking neurons and bistable synapses with spike–timing dependent plasticity. *IEEE Transactions on Neural Networks*, 17(1):211–221, Jan 2006.

[6] J. Arthur and K. Boahen. Learning in silicon: Timing is everything. In Y. Weiss, B. Schölkopf, and J. Platt, editors, *Advances in Neural Information Processing Systems 18*. MIT Press, Cambridge, MA, 2006.

[7] D. Badoni, M. Giulioni, V. Dante, and P. Del Giudice. An aVLSI recurrent network of spiking neurons with reconfigurable and plastic synapses. In *Proceedings of the IEEE International Symposium on Circuits and Systems*, pages 1227–1230. IEEE, IEEE, May 2006.

[8] G. Indiveri and S. Fusi. Spike-based learning in VLSI networks of integrate-and-fire neurons. In *Proc. IEEE International Symposium on Circuits and Systems, ISCAS 2007*, pages 3371–3374, 2007.

[9] S. Fusi and L. F. Abbott. Limits on the memory storage capacity of bounded synapses. *Nature Neuroscience*, 10:485–493, 2007.

[10] E. Chicca, P. Lichtsteiner, T. Delbrück, G. Indiveri, and R.J. Douglas. Modeling orientation selectivity using a neuromorphic multi-chip system. In *Proceedings of IEEE International Symposium on Circuits and Systems*, pages 1235–1238. IEEE, 2006.

[11] C. Bartolozzi and G. Indiveri. Synaptic dynamics in analog VLSI. *Neural Computation*, 19:2581–2603, Oct 2007.

[12] K. A. Boahen. Point-to-point connectivity between neuromorphic chips using address-events. *IEEE Transactions on Circuits and Systems II*, 47(5):416–34, 2000.

[13] J. Lazzaro, S. Ryckebusch, M.A. Mahowald, and C.A. Mead. Winner-take-all networks of $O(n)$ complexity. In D.S. Touretzky, editor, *Advances in neural information processing systems*, volume 2, pages 703–711, San Mateo - CA, 1989. Morgan Kaufmann.

[14] T. Fawcett. An introduction to ROC analysis. *Pattern Recognition Letters*, (26):861–874, 2006.

[15] S. Mitra, G. Indiveri, and S. Fusi. Robust classification of correlated patterns with a neuromorphic VLSI network of spiking neurons. In *IEEE Proceedings on Biomedical Circuits and Systems (BioCAS08)*, 2008. (In press).

